# Characterizing neural dependencies
# with copula models

**Pietro Berkes**
Volen Center for Complex Systems
Brandeis University, Waltham, MA 02454
`berkes@brandeis.edu`

**Frank Wood and Jonathan Pillow**
Gatsby Computational Neuroscience Unit, UCL
London WC1N 3AR, UK
`{fwood,pillow}@gatsby.ucl.ac.uk`

## Abstract

The coding of information by neural populations depends critically on the statistical dependencies between neuronal responses. However, there is no simple model that can simultaneously account for (1) marginal distributions over single-neuron spike counts that are discrete and non-negative; and (2) joint distributions over the responses of multiple neurons that are often strongly dependent. Here, we show that both marginal and joint properties of neural responses can be captured using copula models. Copulas are joint distributions that allow random variables with arbitrary marginals to be combined while incorporating arbitrary dependencies between them. Different copulas capture different kinds of dependencies, allowing for a richer and more detailed description of dependencies than traditional summary statistics, such as correlation coefficients. We explore a variety of copula models for joint neural response distributions, and derive an efficient maximum likelihood procedure for estimating them. We apply these models to neuronal data collected in macaque pre-motor cortex, and quantify the improvement in coding accuracy afforded by incorporating the dependency structure between pairs of neurons. We find that more than one third of neuron pairs shows dependency concentrated in the lower or upper tails for their firing rate distribution.

## 1 Introduction

An important problem in systems neuroscience is to develop flexible, statistically accurate models of neural responses. The stochastic spiking activity of individual neurons in cortex is often well described by a Poisson distribution. Responses from multiple neurons also exhibit strong dependencies (i.e., correlations) due to shared input noise and lateral network interactions. However, there is no natural multivariate generalization of the Poisson distribution. For this reason, much of the literature on population coding has tended either to ignore correlations entirely, treating neural responses as independent Poisson random variables [1, 2], or to adopt a Gaussian model of joint responses [3, 4], assuming a parametric form for dependencies but ignoring key features (e.g., discreteness, non-negativity) of the marginal distribution. Recent work has focused on the construction of large parametric models that capture inter-neuronal dependencies using generalized linear point-process models [5, 6, 7, 8, 9] and binary second-order maximum-entropy models [10, 11, 12]. Although these approaches are quite powerful, they model spike trains only in very fine time bins, and thus describe the dependencies in neural spike count distributions only implicitly.

Modeling the joint distribution of neural activities is therefore an important open problem. Here we show how to construct non-independent joint distributions over firing rates using copulas. In particular, this approach can be used to combine arbitrary marginal firing rate distributions. The development of the paper is as follows: in Section 2, we provide a basic introduction to copulas; in Section 3, we derive a maximum likelihood estimation procedure for neural copula models, in Sections 4 and 5, we apply these models to physiological data collected in macaque pre-motor

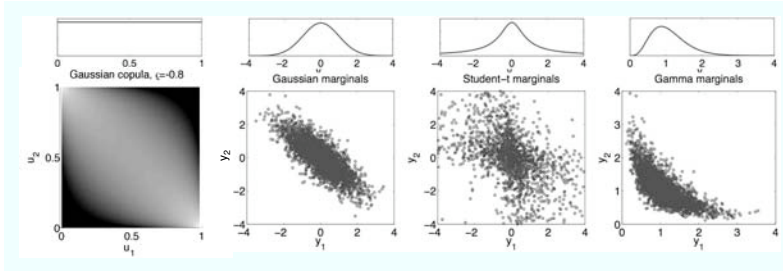

**Figure 1:** Samples drawn from a joint distribution defined using the dependency structure of a bivariate Gaussian distribution and changing the marginal distributions. Top row: The marginal distributions (the leftmost marginal is uniform, by definition of copula). Bottom row: The log-density function of a Gaussian copula, and samples from the joint distribution defined as in Eq. 2.

cortex; finally, in Section 6 we review the insights provided by neural copula models and discuss several extensions and future directions.

## 2 Copulas

A *copula* $C(u_1, \ldots, u_n) : [0, 1]^n \rightarrow [0, 1]$ is a multivariate distribution function on the unit cube with uniform marginals [13, 14]. The basic idea behind copulas is quite simple, and is closely related to that of *histogram equalization*: for a random variable $y_i$ with continuous cumulative distribution function (cdf) $F_i$, the random variable $u_i := F_i(y_i)$ is uniformly distributed on the interval $[0, 1]$. One can use this basic property to separate the marginals from the dependency structure in a multivariate distribution: the full multivariate distribution is standardized by projecting each marginal onto one axis of the unit hyper-cube, and leaving one with a distribution on the hyper-cube (the copula, by definition) that represent dependencies in the marginals' quantiles. This intuition has been formalized in Sklar's Theorem [15]:

**Theorem 1 (Sklar, 1959)** Given $u_1, \ldots, u_n$ random variables with continuous distribution functions $F_1, \ldots, F_n$ and joint distribution $F$, there exist a unique copula $C$ such that for all $u_i$:

$$C(u_1, \ldots, u_n) = F(F_1^{-1}(u_1), \ldots, F_n^{-1}(u_n)) \tag{1}$$

Conversely, given any distribution functions $F_1, \ldots, F_n$ and copula $C$,

$$F(y_1, \ldots, y_n) = C(F_1(y_1), \ldots, F_n(y_n)) \tag{2}$$

is a $n$-variate distribution function with marginal distribution functions $F_1, \ldots, F_n$.

This result gives a way to derive a copula given the joint and marginal distributions (using Eq. 1), and also, more importantly here, to construct a joint distribution by specifying the marginal distributions and the dependency structure separately (Eq. 2). For example, one can keep the dependency structure fixed and vary the marginals (Fig. 1), or vice versa given fixed marginal distributions define new joint distributions using parametrized copula families (Fig. 2). For illustration, in this paper we are going to consider only the bivariate case. All the methods, however, generalize straightforwardly to the multivariate case.

Since copulas do not depend on the marginals, one can define in this way dependency measures that are insensitive to non-linear transformations of the individual variables [14] and generalize correlation coefficients, which are only appropriate for elliptic distributions. The copula representation has also been used to estimate the conditional entropy of neural latencies by separating the contribution of the individual latencies from that coming from their correlations [16].

Dependencies structures are specified by parametric copula families. One notable example is the *Gaussian copula*, which generalizes the dependency structure of the multivariate Gaussian distribution to arbitrary marginal distribution (Fig. 1), and is defined as

$$C(u_1, u_2; \mathbf{\Sigma}) = \Phi_{\mathbf{\Sigma}} \left( \phi^{-1}(u_1), \phi^{-1}(u_2) \right) , \tag{3}$$

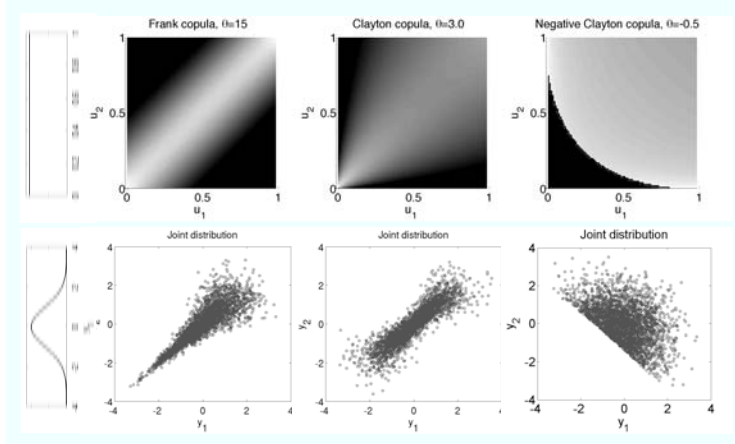

**Figure 2:** Samples drawn from a joint distribution with fixed Gaussian marginals and dependency structure defined by parametric copula families, as indicated by the labels. Top row: log-density function for three copula families. Bottom row: Samples from the joint distribution (Eq. 2).

| | |
|---|---|
| Gaussian | $C^N_{\boldsymbol{\Sigma}}(u_1, u_2) = \Phi_{\boldsymbol{\Sigma}}\left(\phi^{-1}(u_1), \phi^{-1}(u_2)\right)$ |
| Frank | $C^{Fr}_{\theta}(u_1, u_2) = -\frac{1}{\theta} \log\left(1 + \frac{(e^{-\theta u_1}-1)(e^{-\theta u_2}-1)}{e^{-\theta}-1}\right)$ |
| Clayton | $C^{Cl}_{\theta}(u_1, u_2) = (u_1^{-\theta} + u_2^{-\theta} - 1)^{-1/\theta},\ \theta > 0$ |
| Clayton negative | $C^{Neg}_{\theta}(u_1, u_2) = \max\left\{(u_1^{-\theta} + u_2^{-\theta} - 1), 0\right\}^{-1/\theta},\ -1 \leq \theta < 0$ |
| Gumbel | $C^{Gu}_{\theta}(u_1, u_2) = \exp\left(-(\tilde{u}_1^{\theta} + \tilde{u}_2^{\theta})^{1/\theta}\right),\ \tilde{u}_j = -\log u_j,\ \theta \geq 1$ |

**Table 1:** Definition of families of copula distribution functions.

where $\phi(u)$ is the cdf of the univariate Gaussian with mean 0 and variance 1, and $\Phi_{\boldsymbol{\Sigma}}$ is the cdf of a standard multivariate Gaussian with mean $\mathbf{0}$ and covariance matrix $\boldsymbol{\Sigma}$. Other families derive from the economics literature, and are typically one-parameter families that capture various possible dependencies, for example dependencies only in one of the tails of the distribution. Table 1 shows the definition of the copula distributions used in this paper (see [14], for an overview of known copulas and copula construction methods).

## 3 Maximum Likelihood estimation for discrete marginal distributions

In the case where the random variables have discrete distribution functions, as in the case of neural firing rates, only a weaker version of Theorem 1 is valid: there always exists a copula that satisfies Eq. 2, but it is no longer guaranteed to be unique [17]. With discrete data, the probability of a particular outcome is determined by an integral over the region of $[0, 1]^n$ corresponding to that outcome; any two copulas that integrate to the same values on all such regions produce the same joint distribution.

We can derive a Maximum Likelihood (ML) estimation of the parameters $\theta$ by considering a generative model where uniform marginals are generated from the copula density, and in turn use these to generate the discrete variables deterministically using the inverse (marginal) distribution functions, as in Fig. 3. These marginals can be given by the empirical cumulative distribution of firing rates (as in this paper) or by any parametrized family of univariate distributions (such as Poisson).

The ML equation then becomes

$$\underset{\theta}{\mathrm{argmax}}\ p(\mathbf{y}|\theta) = \underset{\theta}{\mathrm{argmax}}\ \int p(\mathbf{y}|\mathbf{u})p(\mathbf{u}|\theta)\mathrm{d}\mathbf{u} \tag{4}$$

$$= \underset{\theta}{\mathrm{argmax}}\ \int_{F_1(y_1-1)}^{F_1(y_1)} \cdots \int_{F_n(y_n-1)}^{F_n(y_n)} c_{\theta}(u_1, \ldots, u_n)\,\mathrm{d}\mathbf{u}\ , \tag{5}$$

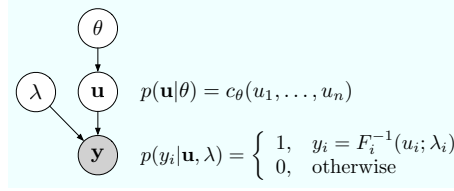

**Figure 3:** Graphical representation of the copula model with discrete marginals. Uniform marginals **u** are drawn from the copula density function $c_\theta(u_1, \ldots, u_n)$, parametrized by $\theta$. The discrete marginals are then generated deterministically using the inverse cdf of the marginals, which are parametrized by $\lambda$.

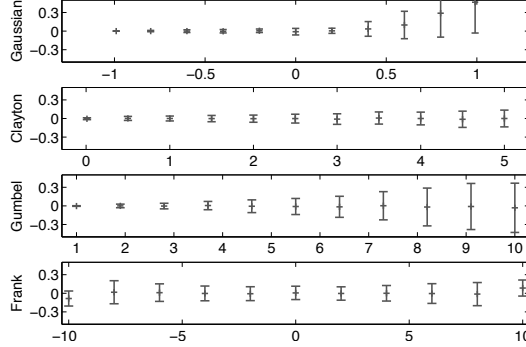

**Figure 4:** Distribution of the maximum likelihood estimation of the parameters of four copula families, for various setting of their parameter (x-axis). On the y-axis, estimates are centered such that 0 corresponds to an unbiased estimate. Error bars are one standard deviation of the estimate.

where $F_i$ can depend on additional parameters $\lambda_i$. The last equation is the copula probability mass inside the volume defined by the vertices $F_i(y_i)$ and $F_i(y_i - 1)$, and can be readily computed using the copula distribution $C_\theta(u_1, \ldots, u_n)$. For example, in the bivariate case one obtains

$$\underset{\theta}{\operatorname{argmax}}\, p(y_1, y_2 | \theta) = \underset{\theta}{\operatorname{argmax}}\, \left[ C_\theta(u_1, u_2) + C_\theta(u_1^-, u_2^-) - C_\theta(u_1^-, u_2) - C_\theta(u_1, u_2^-) \right], \quad (6)$$

where $u_i = F_i(y_i)$ and $u_i^- = F_i(y_i - 1)$.

ML optimization can be performed using standard methods, like gradient descent. In the bivariate case, we find that optimization using the standard MATLAB optimization routines is relatively efficient. Given neural data in the form of firing rates $y_1, y_2$ from a pair of neurons, we collect the empirical cumulative histogram of responses, $F_i(k) = P(y_i \leq k)$. The data is then transformed through the cdfs $u_i = F_i(y_i)$, and the copula model is fit according to Eq. 6. If a parametric distribution family is used for the marginals, the parameters of the copula $\theta$ and those of the marginals $\lambda$ can be estimated simultaneously, or alternatively $\lambda$ can be fitted first, followed by $\theta$. In our experience, the second method is much faster and the quality of the fit is typically unchanged.

We checked for biases in ML estimation due to a limited amount of data and low firing rate by generating data from the discrete copula model (Fig. 3), for a number of copula families and Poisson marginals with parameters $\lambda_1 = 2, \lambda_2 = 3$. The estimate is based on 3500 observations generated from the models (1000 for the Gaussian copula). The estimation is repeated 200 times (100 for the Gaussian copula) in order to compute the mean and standard deviation of the ML estimate. Figure 4 shows that the estimate is unbiased and accurate for a wide range of parameters. Inaccuracy in the estimation becomes larger as the copulas approach functional dependency (i.e., $u_2 = f(u_1)$ for a deterministic function $f$), as it is the case for the Gaussian copula when $\rho$ tends to 1, and for the Gumbel copula as $\theta$ goes to infinity. This is an effect due to low firing rates. Given our formulation of the estimation problem as a generative model, one could use more sophisticated Bayesian methods in place of the ML estimation, in order to infer a whole distribution over parameters given the data. This would allow to put error bars on the estimated parameters, and would avoid overfitting at the cost of computational time.

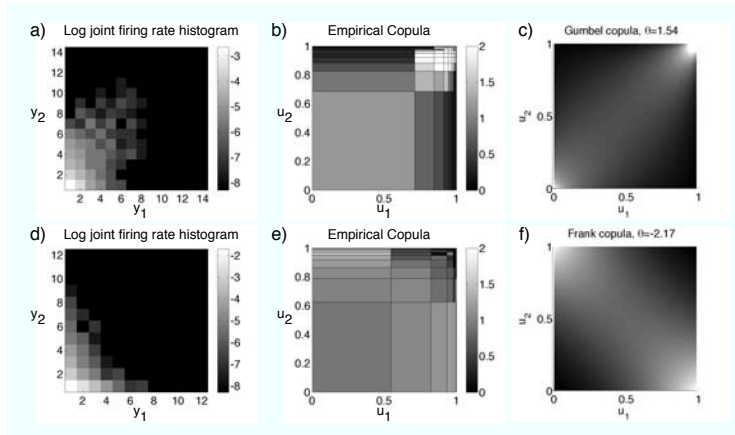

**Figure 5:** Empirical joint distribution and copula fit for two neuron pairs. The top row shows two neurons that have dependencies mainly in the upper tails of their marginal distribution. The pair in the bottom row has negative dependency. a,d) Histogram of the firing rate of the two neurons. Colors correspond to the logarithm of the normalized frequency. b,e) Empirical copula. The color intensity has been cut off at 2.0 to improve visibility. c,f) Density of the copula fit.

## 4  Results

To demonstrate the ability of copula models to fit joint firing rate distribution, we model neural data recorded using a multi-electrode array implanted in the pre-motor cortex (PMd) area of a macaque monkey [18, 19]. The array consisted in $10 \times 10$ electrodes separated by $400\mu$m. Firing times were recorded while the monkey executed a center-out reaching task. See [19] for a description of the task and general experimental setup. We fit the copula model using the marginal distribution of neural activity over the entire recording session, including data recorded between trials (i.e., while the monkey was freely behaving). Although one might also like to consider data collected during a single task condition (i.e., the stimulus-conditional response distribution), the marginal response distribution is an important statistical object in its own right, and has been the focus of recent much literature [10, 11]. For example, the joint activity across neurons, averaged over stimuli, is the only distribution the brain has access to, and must be sufficient for learning to construct representations of the external world.

We collected spike responses in 100ms bins, and selected at random, without repetition, a training set of 4000 bins and a test set of 2000 bins. Out of a total of 194 neurons we select a subset of 33 neurons that fired a minimum of 2500 spikes over the whole data set. For every pair of neurons in this subset (528 pairs), we fit the parameters of several copula families to the joint firing rate.

Figure 5 shows two examples of the kind of the dependencies present in the data set and how they are fit by different copula families. The neuron pair in the top row shows dependency in the upper tails of their distribution, as can be seen in the histogram of joint firing rates (colors represent the logarithm of the frequency): The two neurons have the tendency to fire strongly together, but are relatively independent at low firing rates. This is confirmed by the *empirical copula*, which shows the probability mass in the regions defined by the cdfs of the marginal distribution. Since the marginal cdfs are discrete, the data is projected on a discrete set of points on the unit cube; the colors in the empirical copula plots represent the probability mass in the region where the marginal cdfs are constant. The axis in the empirical copula should be interpreted as the quantiles of the marginal distributions – for example, 0.5 on the x-axis corresponds to the median of the distribution of $y_1$. The higher probability mass in the upper right corner of the plot thus means that the two neurons tend to be in the upper tails of the distributions simultaneously, and thus to have higher firing rates together. On the right, one can see that this dependency structure is well captured by the Gumbel copula fit. The second pair of neuron in the bottom row have negative dependency, in the sense that when one of them has high firing rate the other tends to be silent. Although this is not readily visible in the joint histogram, the dependency becomes clear in the empirical copula plot. This structure is captured by the Frank copula fit.

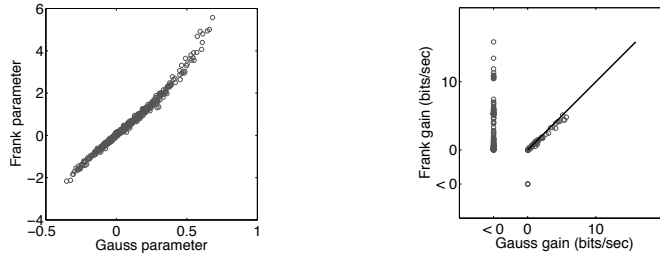

**Figure 6:** In the pairs where their fit improves over the independence model, the parameters (left) and the score (right) of the Gaussian and Frank models are highly correlated.

The goodness-of-fit of the copula families is evaluated by cross-validation: We fit different models on training data, and compute the log-likelihood of test data under the fitted model. The models are scored according to the difference between the log-likelihood of a model that assumes independent neurons and the log-likelihood of the copula model. This measure (appropriately renormalized) can be interpreted as the number of bits per second that can be saved when coding the firing rate by taking into account the dependencies encoded by the copula family. This is because this quantity can be expressed as an estimation of the difference in the Kullback-Leibler divergence of the independent ($p_{indep}$) and copula model ($p_\theta$) to the real distribution $p^*$

$$\langle \log p_\theta(y) \rangle_{y \sim p^*} - \langle \log p_{indep}(y) \rangle_{y \sim p^*} \tag{7}$$

$$\approx \int p^*(y) \log p_\theta(y) dy - \int p^*(y) \log p_{indep}(y) \tag{8}$$

$$= KL(p^* || p_{indep}) - KL(p^* || p_\theta). \tag{9}$$

We took particular care in selecting a small set of copula families that would be able to capture the dependencies occurring in the data. Some of the families that we considered at first capture similar kind of dependencies, and their scores are highly correlated. For example, the Frank and Gaussian copulas are able to represent both positive and negative dependencies in the data, and simultaneously in lower and upper tails, although the dependencies in the tails are less strong for the Frank family (compare the copula densities in Figs. 1 and 5f). Fig. 6 (left) shows that both the parameter fits and their performance are highly correlated. An advantage of the Frank copula is that it is much more efficient to fit, since the Gaussian copula requires multiple evaluations of the bivariate Gaussian cdf, which requires expensive numerical calculations. In addition, The Gaussian copula was also found to be more prone to overfitting on this data set (Fig. 6, right). For these reasons, we decided to use the Frank family only for the rest of the analysis.

With similar procedures we shortlisted a total 3 families that cover the vast majority of dependencies in our data set: Frank, Clayton, and Gumbel copulas. Examples of the copula density of these families can be found in Figs. 2, and 5. The Clayton and Gumbel copulas describe dependencies in the lower and upper tails of the distributions, respectively. We didn't find any example of neuron pairs where the dependency would be in the upper tail of the distribution for one and in the lower tail for the other distribution, or more complicated dependencies.

Out of all 528 neuron pairs, 393 had a significant improvement (P<0.05 on test data) over a model with independent neurons[1] and for 102 pairs the improvement was larger than 1 bit/sec. Dependencies in the data set seem thus to be widespread, despite the fact that individual neurons are recorded from electrodes that are up to 4.4 mm apart. Fig. 7 shows the histogram of improvement in bits/sec. The most common dependencies structures over all neuron pairs are given by the Gaussian-like dependencies of the Frank copula ($54\%$ of the pairs). Interestingly, a large proportion of the neurons showed dependencies concentrated in the upper tails (Gumbel copula, $22\%$) or lower tails (Clayton copula, $16\%$) of the distributions (Fig. 7).

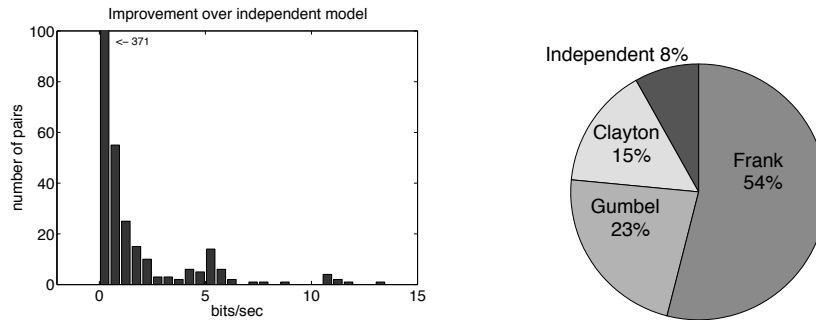

**Figure 7:** For every pair of neurons, we select the copula family that shows the largest improvement over a model with independent neurons, in bits/sec. Left: histogram of the gain in bits/sec over the independent model. Right: Pie chart of the copula families that best fit the neuron pairs.

## 5    Discussion

The results presented here show that it is possible to represent neuronal spike responses using a model that preserves discrete, non-negative marginals while incorporating various types of dependencies between neurons. Mathematically, it is straightforward to generalize these methods to the $n$-variate case (i.e., distributions over the responses of $n$ neurons). However, many copula families have only one or two parameters, regardless of the copula dimensionality. If the dependency structure across a neural population is relatively homogeneous, then these copulas may be useful in that they can be estimated using far less data than required, e.g., for a full covariance matrix (which has $O(n^2)$ parameters). On the other hand, if the dependencies within a population vary markedly for different pairs of neurons (as in the data set examined here), such copulas will lack the flexibility to capture the complicated dependencies within a full population. In such cases, we can still apply the Gaussian copula (and other copulas derived from elliptically symmetric distributions), since it is parametrized by the same covariance matrix as a $n$-dimensional Gaussian. However, the Gaussian copula becomes prohibitively expensive to fit in high dimensions, since evaluating the likelihood requires an exponential number of evaluations of the multivariate Gaussian cdf, which itself must be computed numerically.

One challenge for future work will therefore be to design new parametric families of copulas whose parameters grow with the number of neurons, but remain tractable enough for maximum-likelihood estimation. Recently, Kirshner [20] proposed a copula-based representation for multivariate distributions using a model that averages over tree-structured copula distributions. The basic idea is that pairwise copulas can be easily combined to produce a tree-structured representation of a multivariate distribution, and that averaging over such trees gives an even more flexible class of multivariate distributions. We plan to examine this approach using neural population data in future work.

Another future challenge is to combine explicit models of the stimulus-dependence underlying neural responses with models capable of capturing their joint response dependencies. The data set analyzed here concerned the distribution over spike responses during all all stimulus conditions (i.e., the marginal distribution over responses, as opposed to the the conditional response distribution given a stimulus). Although this marginal response distribution is interesting in its own right, for many applications one is interested in separating correlations that are induced by external stimuli from internal correlations due to the network interactions. One obvious approach is to consider a hybrid model with a Linear-Nonlinear-Poisson model [21] capturing stimulus-induced correlation, adjoined to a copula distribution that models the residual dependencies between neurons (Fig. 8). This is an important avenue for future exploration.

**Acknowledgments**

We'd like to thank Matthew Fellows for providing the data used in this study. This work was supported by the Gatsby Charitable Foundation.

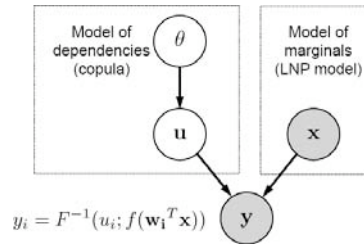

$$y_i = F^{-1}(u_i; f(\mathbf{w_i}^T \mathbf{x}))$$

**Figure 8:** Hybrid LNP-copula model. The LNP part of the model removes stimulus-induced correlations from the neural data, so that the copula model can take into account residual network-related dependencies.

## Footnotes

[1]We computed the significance level by generating an artificial data set using independent neurons with the same empirical pdf as the monkey data. We analyzed the generated data and computed the maximal improvement over an independent model (due to the limited number of samples) on artificial test data. The resulting distribution is very narrowly distributed around zero. We took the 95th percentile of the distribution (0.02 bits/sec) as the threshold for significance.

## References

[1] R. Zemel, P. Dayan, and A. Pouget. Probabilistic interpretation of population codes. *Neural Computation*, 10:403–430, 1998.

[2] A. Pouget, K. Zhang, S. Deneve, and P.E. Latham. Statistically efficient estimation using population coding. *Neural Computation*, 10(2):373–401, 1998.

[3] L. Abbott and P. Dayan. The effect of correlated variability on the accuracy of a population code. *Neural Computation*, 11:91–101, 1999.

[4] E. Maynard, N. Hatsopoulos, C. Ojakangas, B. Acuna, J. Sanes, R. Normann, and J. Donoghue. Neuronal interactions improve cortical population coding of movement direction. *Journal of Neuroscience*, 19:8083–8093, 1999.

[5] E. Chornoboy, L. Schramm, and A. Karr. Maximum likelihood identification of neural point process systems. *Biological Cybernetics*, 59:265–275, 1988.

[6] W. Truccolo, U. T. Eden, M. R. Fellows, J. P. Donoghue, and E. N. Brown. A point process framework for relating neural spiking activity to spiking history, neural ensemble and extrinsic covariate effects. *J. Neurophysiol*, 93(2):1074–1089, 2004.

[7] M. Okatan, M. Wilson, and E. Brown. Analyzing functional connectivity using a network likelihood model of ensemble neural spiking activity. *Neural Computation*, 17:1927–1961, 2005.

[8] S. Gerwinn, J.H. Macke, M. Seeger, and M. Bethge. Bayesian inference for spiking neuron models with a sparsity prior. *Advances in Neural Information Processing Systems*, 2008.

[9] J. W. Pillow, J. Shlens, L. Paninski, A. Sher, A. M. Litke, and E. P. Chichilnisky, E. J. Simoncelli. Spatio-temporal correlations and visual signaling in a complete neuronal population. *Nature*, 454(7206):995–999, 2008.

[10] E. Schneidman, M. Berry, R. Segev, and W. Bialek. Weak pairwise correlations imply strongly correlated network states in a neural population. *Nature*, 440:1007–1012, 2006.

[11] J. Shlens, G. Field, J. Gauthier, M. Grivich, D. Petrusca, A. Sher, Litke A. M., and E. J. Chichilnisky. The structure of multi-neuron firing patterns in primate retina. *J Neurosci*, 26:8254–8266, 2006.

[12] J.H. Macke, P. Berens, A.S. Ecker, A.S. Tolias, and M. Bethge. Generating spike trains with specified correlation coefficients. *Neural Computation*, 21(2), 2009.

[13] H. Joe. *Multivariate models and dependence concepts*. Chapman & Hall, London, 1997.

[14] R.B. Nelsen. *An introduction to copulas*. Springer, New York, 2nd edition, 2006.

[15] A. Sklar. Fonctions de répartition à *n* dimensions et leurs marges. *Publ. Inst. Statist. Univ. Paris*, 8:229–231, 1959.

[16] R.L. Jenison and R.A. Reale. The shape of neural dependence. *Neural Computation*, 16(4):665–672, 2004.

[17] C. Genest and J. Neslehova. A primer on copulas for count data. *Astin Bulletin*, 37(2):475–515, 2007.

[18] M. Serruya, N. Hatsopoulos, L. Paninski, M. Fellows, and J. Donoghue. Instant neural control of a movement signal. *Nature*, 416:141–142, 2002.

[19] S. Suner, MR Fellows, C. Vargas-Irwin, GK Nakata, and JP Donoghue. Reliability of signals from a chronically implanted, silicon-based electrode array in non-human primate primary motor cortex. *Neural Systems and Rehabilitation Engineering, IEEE Transactions on*, 13(4):524–541, 2005.

[20] S. Kirshner. Learning with tree-averaged densities and distributions. *NIPS*, 20, 2008.

[21] E. P. Simoncelli, L. Paninski, J. Pillow, and O. Schwartz. Characterization of neural responses with stochastic stimuli. In M. Gazzaniga, editor, *The Cognitive Neurosciences*, pages 327–338. MIT Press, 3rd edition, 2004.

